# Radial Basis Function Network for Multi-task Learning

**Xuejun Liao**
Department of ECE
Duke University
Durham, NC 27708-0291, USA
xjliao@ee.duke.edu

**Lawrence Carin**
Department of ECE
Duke University
Durham, NC 27708-0291, USA
lcarin@ee.duke.edu

## Abstract

We extend radial basis function (RBF) networks to the scenario in which multiple correlated tasks are learned simultaneously, and present the corresponding learning algorithms. We develop the algorithms for learning the network structure, in either a supervised or unsupervised manner. Training data may also be actively selected to improve the network's generalization to test data. Experimental results based on real data demonstrate the advantage of the proposed algorithms and support our conclusions.

## 1 Introduction

In practical applications, one is frequently confronted with situations in which multiple tasks must be solved. Often these tasks are not independent, implying what is learned from one task is transferable to another correlated task. By making use of this transferability, each task is made easier to solve. In machine learning, the concept of explicitly exploiting the transferability of expertise between tasks, by learning the tasks simultaneously under a unified representation, is formally referred to as "multi-task learning" [1].

In this paper we extend radial basis function (RBF) networks [4,5] to the scenario of multi-task learning and present the corresponding learning algorithms. Our primary interest is to learn the regression model of several data sets, where any given data set may be correlated with some other sets but not necessarily with all of them. The advantage of multi-task learning is usually manifested when the training set of each individual task is weak, i.e., it does not generalize well to the test data. Our algorithms intend to enhance, in a mutually beneficial way, the weak training sets of multiple tasks, by learning them simultaneously. Multi-task learning becomes superfluous when the data sets all come from the same generating distribution, since in that case we can simply take the union of them and treat the union as a single task. In the other extreme, when all the tasks are independent, there is no correlation to utilize and we learn each task separately.

The paper is organized as follows. We define the structure of multi-task RBF network in Section 2 and present the supervised learning algorithm in Section 3. In Section 4 we show how to learn the network structure in an unsupervised manner, and based on this we demonstrate how to actively select the training data, with the goal of improving the

generalization to test data. We perform experimental studies in Section 5 and conclude the paper in Section 6.

## 2 Multi-Task Radial Basis Function Network

Figure 1 schematizes the radial basis function (RBF) network structure customized to multitask learning. The network consists of an input layer, a hidden layer, and an output layer. The input layer receives a data point $\mathbf{x} = [x_1, \cdots, x_d]^T \in \mathbb{R}^d$ and submits it to the hidden layer. Each node at the hidden layer has a localized activation $\phi^n(\mathbf{x}) = \phi(||\mathbf{x} - \mathbf{c}_n||, \sigma_n)$, $n = 1, \cdots, N$, where $|| \cdot ||$ denotes the vector norm and $\phi^n(\cdot)$ is a radial basis function (RBF) localized around $\mathbf{c}_n$ with the degree of localization parameterized by $\sigma_n$. Choosing $\phi(z, \sigma) = \exp(-\frac{z^2}{2\sigma^2})$ gives the Gaussian RBF. The activations of all hidden nodes are weighted and sent to the output layer. Each output node represents a unique task and has its own hidden-to-output weights. The weighted activations of the hidden nodes are summed at each output node to produce the output for the associated task. Denoting $\mathbf{w}_k = [w_{0k}, w_{1k}, \cdots, w_{Nk}]^T$ as the weights connecting hidden nodes to the $k$-th output node, then the output for the $k$-th task, in response to input $\mathbf{x}$, takes the form

$$f_k(\mathbf{x}) = \mathbf{w}_k^T \boldsymbol{\phi}(\mathbf{x}) \tag{1}$$

where $\boldsymbol{\phi}(\mathbf{x}) = \left[\phi^0(\mathbf{x}), \phi^1(\mathbf{x}), \ldots, \phi^N(\mathbf{x})\right]^T$ is a column containing $N + 1$ basis functions with $\phi^0(\mathbf{x}) \equiv 1$ a dummy basis accounting for the bias in Figure 1.

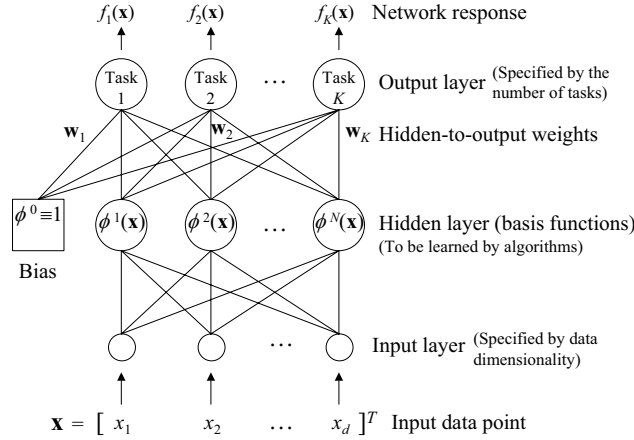

Figure 1: A multi-task structure of RBF Network. Each of the output nodes represents a unique task. Each task has its own hidden-to-output weights but all the tasks share the same hidden nodes. The activation of hidden node $n$ is characterized by a basis function $\phi_n(\mathbf{x}) = \phi(||\mathbf{x} - \mathbf{c}_n||, \sigma_n)$. A typical choice of $\phi$ is $\phi(z, \sigma) = \exp(-\frac{z^2}{2\sigma^2})$, which gives the Gaussian RBF.

## 3 Supervised Learning

Suppose we have $K$ tasks and the data set of the $k$-th task is $\mathcal{D}_k = \{(\mathbf{x}_{1k}, y_{1k}), \cdots, (\mathbf{x}_{J_k k}, y_{J_k k})\}$, where $y_{ik}$ is the target (desired output) of $\mathbf{x}_{ik}$. By definition, a given data point $\mathbf{x}_{ik}$ is said to be supervised if the associated target $y_{ik}$ is provided and unsupervised if $y_{ik}$ is not provided. The definition extends similarly to a set of data

Table 1: Learning Algorithm of Multi-Task RBF Network

| |
|---|
| Input: $\{(\mathbf{x}_{1k}, y_{2k}), \cdots, (\mathbf{x}_{J_k,k}, y_{J_k,k})\}_{k=1:K}$, $\phi(\cdot, \sigma)$, $\sigma$, and $\rho$;     Output: $\phi(\cdot)$ and $\{\mathbf{w}_k\}_{k=1}^K$. |

1. **For** $m=1\!:\!K$, **For** $n=1\!:\!J_m$, **For** $k=1\!:\!K$, **For** $i=1\!:\!J_k$
     Compute $\widehat{\phi}_{ik}^{nm} = \phi(||\mathbf{x}_{nm} - \mathbf{x}_{ik}||, \sigma)$;
2. **Let** $N = 0$, $\phi(\cdot) = 1$, $e_0 = \sum_{k=1}^K \left[ \sum_{i=1}^{J_k} y_{ik}^2 - (J_k + \rho)^{-1} (\sum_{i=1}^{J_k} y_{ik})^2 \right]$;
     **For** $k=1\!:\!K$, compute $\mathbf{A}_k = J_k + \rho$, $\mathbf{w}_k = (J_k + \rho)^{-1} \sum_{i=1}^{J_k} y_{ik}$;
3. **For** $m=1\!:\!K$, **For** $n=1\!:\!J_m$
     **If** $\widehat{\phi}^{nm}$ is not marked as "deleted"
         **For** $k=1\!:\!K$, compute
             $\mathbf{c}_k = \sum_{i=1}^{J_k} \phi_{ik} \widehat{\phi}_{ik}^{nm}$,     $q_k = \sum_{i=1}^{J_k} (\widehat{\phi}_{ik}^{nm})^2 + \rho - \mathbf{c}_k^T \mathbf{A}_k^{-1} \mathbf{c}_k$;
         **If** there exists $k$ such that $q_k = 0$, mark $\widehat{\phi}^{nm}$ as "deleted";
         **else**, compute $\delta e(\phi, \widehat{\phi}^{nm})$ using (5).
4. **If** $\{\widehat{\phi}^{ik}\}_{i=1:J_k, k=1:K}$ are all marked as "deleted", go to 10.
5. **Let** $(n^*, m^*) = \arg \max_{\widehat{\phi}^{nm} \text{ not marked as "deleted"}} \delta e(\phi, \widehat{\phi}^{nm})$; Mark $\widehat{\phi}^{n^* m^*}$ as "deleted".
6. Tune RBF parameter $\sigma_{N+1} = \arg \max_\sigma \delta e(\phi, \phi(|| \cdot - \mathbf{x}_{n^* m^*}||, \sigma))$
7. **Let** $\phi^{N+1}(\cdot) = \phi(|| \cdot - \mathbf{x}_{n^* m^*}||, \sigma_{N+1})$; Update $\phi(\cdot) \leftarrow [\phi^T(\cdot), \phi^{N+1}(\cdot)]^T$;
8. **For** $k=1\!:\!K$
     Compute $\mathbf{A}_k^{new}$ and $\mathbf{w}_k^{new}$ respectively by (A-1) and (A-3) in the appendix; Update $\mathbf{A}_k \leftarrow \mathbf{A}_k^{new}$, $\mathbf{w}_k \leftarrow \mathbf{w}_k^{new}$
9. **Let** $e_{N+1} = e_N - \delta e(\phi, \phi^{N+1})$;    **If** the sequence $\{e_n\}_{n=0:(N+1)}$ is converged, go to 10, **else** update $N \leftarrow N + 1$ and go back to 3.
10. **Exit** and output $\phi(\cdot)$ and $\{\mathbf{w}_k\}_{k=1}^K$.

points. We are interested in learning the functions $f_k(\mathbf{x})$ for the $K$ tasks, based on $\cup_{k=1}^K \mathcal{D}_k$. The learning is based on minimizing the squared error

$$e(\phi, \mathbf{w}) = \sum_{k=1}^K \left\{ \sum_{i=1}^{J_k} \left(\mathbf{w}_k^T \phi_{ik} - y_{ik}\right)^2 + \rho ||\mathbf{w}_k||^2 \right\} \qquad (2)$$

where $\phi_{ik} = \phi(\mathbf{x}_{ik})$ for notational simplicity. The regularization terms $\rho ||\mathbf{w}_k||^2$, $k = 1, \cdots, K$, are used to prevent singularity of the $\mathbf{A}$ matrices defined in (3), and $\rho$ is typically set to a small positive number. For fixed $\phi$'s, the $\mathbf{w}$'s are solved by minimizing $e(\phi, \mathbf{w})$ with respect to $\mathbf{w}$, yielding

$$\mathbf{w}_k = \mathbf{A}_k^{-1} \sum_{i=1}^{J_k} y_{ik} \phi_{ik} \quad \text{and} \quad \mathbf{A}_k = \sum_{i=1}^{J_k} \phi_{ik} \phi_{ik}^T + \rho \mathbf{I}, \qquad k = 1, \cdots, K \qquad (3)$$

In a multi-task RBF network, the input layer and output layer are respectively specified by the data dimensionality and the number of tasks. We now discuss how to determine the hidden layer (basis functions $\phi$). Substituting the solutions of the $\mathbf{w}$'s in (3) into (2) gives

$$e(\phi) = \sum_{k=1}^K \sum_{i=1}^{J_k} \left( y_{ik}^2 - y_{ik} \mathbf{w}_k^T \phi_{ik} \right) \qquad (4)$$

where $e(\phi)$ is a function of $\phi$ only because $\mathbf{w}$'s are now functions of $\phi$ as given by (3). By minimizing $e(\phi)$, we can determine $\phi$. Recalling that $\phi_{ik}$ is an abbreviation of $\phi(\mathbf{x}_{ik}) = \left[1, \phi^1(\mathbf{x}_{ik}), \dots, \phi^N(\mathbf{x}_{ik})\right]^T$, this amounts to determining $N$, the number of basis functions, and the functional form of each basis function $\phi^n(\cdot)$, $n = 1, \dots, N$. Consider the candidate functions $\{\phi^{nm}(\mathbf{x}) = \phi(||\mathbf{x} - \mathbf{x}_{nm}||, \sigma) : n = 1, \cdots, J_m, m = 1, \cdots, K\}$. We learn the RBF network structure by selecting $\phi(\cdot)$ from these candidate functions such that $e(\phi)$ in (4) is minimized. The following theorem tells us how to perform the selection in a sequential way; the proof is given in the Appendix.

**Theorem 1** *Let $\phi(\mathbf{x}) = [1, \phi^1(\mathbf{x}), \dots, \phi^N(\mathbf{x})]^T$ and $\phi^{N+1}(\mathbf{x})$ be a single basis function. Assume the $\mathbf{A}$ matrices corresponding to $\phi$ and $[\phi, \phi^{N+1}]^T$ are all non-degenerate. Then*

$$\delta e(\phi, \phi^{N+1}) = e(\phi) - e([\phi, \phi^{N+1}]^T) = \sum_{k=1}^K \left(\mathbf{c}_k^T \mathbf{w}_k - \sum_{i=1}^{J_k} y_{ik} \phi_{ik}^{N+1}\right)^2 q_k^{-1} \quad (5)$$

*where $\phi_{ik}^{N+1} = \phi^{N+1}(\phi_{ik})$, $\mathbf{w}_k$ and $\mathbf{A}$ are the same as in (3), and*

$$\mathbf{c}_k = \sum_{i=1}^{J_k} \phi_{ik} \phi_{ik}^{N+1}, \qquad d_k = \sum_{i=1}^{J_k} (\phi_{ik}^{N+1})^2 + \rho, \qquad q_k = d_k - \mathbf{c}_k^T \mathbf{A}_k^{-1} \mathbf{c}_k \quad (6)$$

By the conditions of the theorem $\mathbf{A}_k^{new}$ is full rank and hence it is positive definite by construction. By (A-2) in the Appendix, $q_k^{-1}$ is a diagonal element of $(\mathbf{A}_k^{new})^{-1}$, therefore $q_k^{-1}$ is positive and by (5) $\delta e(\phi, \phi^{N+1}) > 0$, which means adding $\phi^{N+1}$ to $\phi$ generally makes the squared error decrease. The decrease $\delta e(\phi, \phi^{N+1})$ depends on $\phi^{N+1}$. By sequentially selecting basis functions that bring the maximum error reduction, we achieve the goal of maximizing $e(\phi)$. The details of the learning algorithm are summarized in Table 1.

## 4   Active Learning

In the previous section, the data in $D_k$ are supervised (provided with the targets). In this section, we assume the data in $D_k$ are initially unsupervised (only $\mathbf{x}$ is available without access to the associated $y$) and we select a subset from $\mathcal{D}_k$ to be supervised (targets acquired) such that the resulting network generalizes well to the remaining data in $\mathcal{D}_k$. The approach is generally known as active learning [6]. We first learn the basis functions $\phi$ from the unsupervised data, and based on $\phi$ select data to be supervised. Both of these steps are based on the following theorem, the proof of which is given in the Appendix.

**Theorem 2** *Let there be $K$ tasks and the data set of the $k$-th task is $\mathcal{D}_k \cup \widetilde{\mathcal{D}}_k$ where $\mathcal{D}_k = \{(\mathbf{x}_{ik}, y_{ik})\}_{i=1}^{J_k}$ and $\widetilde{\mathcal{D}}_k = \{(\mathbf{x}_{ik}, y_{ik})\}_{i=J_k+1}^{J_k+\widetilde{J}_k}$. Let there be two multi-task RBF networks, whose output nodes are characterized by $f_k(\cdot)$ and $f_{\widetilde{k}}(\cdot)$, respectively, for task $k = 1, \dots, K$. The two networks have the same given basis functions (hidden nodes) $\phi(\cdot) = [1, \phi^1(\cdot), \cdots, \phi^N(\cdot)]^T$, but different hidden-to-output weights. The weights of $f_k(\cdot)$ are trained with $\mathcal{D}_k \cup \widetilde{\mathcal{D}}_k$, while the weights of $f_{\widetilde{k}}(\cdot)$ are trained using $\widetilde{\mathcal{D}}_k$. Then for $k = 1, \cdots, K$, the square errors committed on $\mathcal{D}_k$ by $f_k(\cdot)$ and $f_{\widetilde{k}}(\cdot)$ are related by*

$$0 \le [det\,\mathbf{\Gamma}_k]^{-1} \le \lambda_{max,k}^{-1} \le \left[\sum_{i=1}^{J_k}(y_{ik} - f_{\widetilde{k}}(\mathbf{x}_{ik}))^2\right]^{-1} \sum_{i=1}^{J_k}(y_{ik} - f_k(\mathbf{x}_{ik}))^2 \le \lambda_{min,k}^{-1} \le 1 \quad (7)$$

*where $\mathbf{\Gamma}_k = \left[\mathbf{I} + \mathbf{\Phi}_k^T(\rho\,\mathbf{I} + \widetilde{\mathbf{\Phi}}_k \widetilde{\mathbf{\Phi}}_k^T)^{-1}\mathbf{\Phi}_k\right]^2$ with $\mathbf{\Phi} = [\phi(\mathbf{x}_{1k}), \dots, \phi(\mathbf{x}_{J_k k})]$ and $\widetilde{\mathbf{\Phi}} = [\phi(\mathbf{x}_{J_k+1,k}), \dots, \phi(\mathbf{x}_{J_k+\widetilde{J}_k,k})]$, and $\lambda_{max,k}$ and $\lambda_{min,k}$ are respectively the largest and smallest eigenvalues of $\mathbf{\Gamma}_k$.*

Specializing Theorem 2 to the case $\widetilde{J}_k = 0$, we have

**Corollary 1** *Let there be $K$ tasks and the data set of the $k$-th task is $\mathcal{D}_k = \{(\mathbf{x}_{ik}, y_{ik})\}_{i=1}^{J_k}$. Let the RBF network, whose output nodes are characterized by $f_k(\cdot)$ for task $k = 1, \dots, K$, have given basis functions (hidden nodes) $\phi(\cdot) = [1, \phi^1(\cdot), \cdots, \phi^N(\cdot)]^T$ and the hidden-to-output weights of task $k$ be trained with $\mathcal{D}_k$. Then for $k = 1, \cdots, K$, the squared error committed on $\mathcal{D}_k$ by $f_k(\cdot)$ is bounded as $0 \le [det\,\mathbf{\Gamma}_k]^{-1} \le \lambda_{max,k}^{-1} \le \left[\sum_{i=1}^{J_k} y_{ik}^2\right]^{-1} \sum_{i=1}^{J_k}(y_{ik} - f_k(\mathbf{x}_{ik}))^2 \le \lambda_{min,k}^{-1} \le 1$, where $\mathbf{\Gamma}_k = (\mathbf{I} + \rho^{-1}\mathbf{\Phi}_k^T \mathbf{\Phi}_k)^2$ with $\mathbf{\Phi} = [\phi(\mathbf{x}_{1,k}), \dots, \phi(\mathbf{x}_{J_k,k})]$, and $\lambda_{max,k}$ and $\lambda_{min,k}$ are respectively the largest and smallest eigenvalues of $\mathbf{\Gamma}_k$.*

It is evident from the properties of matrix determinant [7] and the definition of $\mathbf{\Phi}$ that $\det\mathbf{\Gamma}_k = \left[\det(\rho\mathbf{I} + \mathbf{\Phi}_k\mathbf{\Phi}_k^T)\right]^2 [\det(\rho\,\mathbf{I})]^{-2} = \left[\det(\rho\mathbf{I} + \sum_{i=1}^{J_k}\phi_{ik}\phi_{ik}^T)\right]^2 [\det(\rho\,\mathbf{I})]^{-2}$.

Using (3) we write succinctly $\det \mathbf{\Gamma}_k = [\det \mathbf{A}_k^2][\det(\rho\mathbf{I})]^{-2}$. We are interested in selecting the basis functions $\phi$ that minimize the error, before seeing $y$'s. By Corollary 1 and the equation $\det \mathbf{\Gamma}_k = [\det \mathbf{A}_k^2][\det(\rho\mathbf{I})]^{-2}$, the squared error is lower bounded by $\sum_{i=1}^{J_k} y_{ik}^2 [\det(\rho\mathbf{I})]^2 [\det \mathbf{A}_k]^{-2}$. Instead of minimizing the error directly, we minimize its lower bound. As $[\det(\rho\mathbf{I})]^2 \sum_{i=1}^{L_k} y_{ik}^2$ does not depend on $\phi$, this amounts to selecting $\phi$ to minimize $(\det \mathbf{A}_k)^{-2}$. To minimize the errors for all tasks $k = 1\cdots, K$, we select $\phi$ to minimize $\prod_{k=1}^{K}(\det \mathbf{A}_k)^{-2}$.

The selection proceeds in a sequential manner. Suppose we have selected basis functions $\phi = [1, \phi^1, \cdots, \phi^N]^T$. The associated $\mathbf{A}$ matrices are $\mathbf{A}_k = \sum_{i=1}^{J_k} \phi_{ik}\phi_{ik}^T + \rho\mathbf{I}_{(N+1)\times(N+1)}$, $k = 1, \cdots, K$. Augmenting basis functions to $[\phi^T, \phi^{N+1}]^T$, the $\mathbf{A}$ matrices change to $\mathbf{A}_k^{new} = \sum_{i=1}^{J_k}[\phi_{ik}^T, \phi_{ik}^{N+1}]^T[\phi_{ik}^T, \phi_{ik}^{N+1}] + \rho\mathbf{I}_{(N+2)\times(N+2)}$. Using the determinant formula of block matrices [7], we get $\prod_{k=1}^{K}(\det \mathbf{A}_k^{new})^{-2} = \prod_{k=1}^{K}(q_k \det \mathbf{A}_k)^{-2}$, where $q_k$ is the same as in (6). As $\mathbf{A}_k$ does not depend on $\phi^{N+1}$, the left-hand side is minimized by maximizing $\prod_{k=1}^{K} q_k^2$. The selection is easily implemented by making the following two minor modifications in Table 1: (a) in step 2, compute $e_0 = \sum_{k=1}^{K} \ln(J_k + \rho)^{-2}$; in step 3, compute $\delta e(\phi, \widehat{\phi}^{nm}) = \sum_{k=1}^{K} \ln q_k^2$. Employing the logarithm is for gaining additivity and it does not affect the maximization.

Based on the basis functions $\phi$ determined above, we proceed to selecting data to be supervised and determining the hidden-to-output weights $\mathbf{w}$ from the supervised data using the equations in (3). The selection of data is based on an iterative use of the following corollary, which is a specialization of Theorem 2 and was originally given in [8].

**Corollary 2** *Let there be $K$ tasks and the data set of the $k$-th task is $\mathcal{D}_k = \{(\mathbf{x}_{ik}, y_{ik})\}_{i=1}^{J_k}$. Let there be two RBF networks, whose output nodes are characterized by $f_k(\cdot)$ and $f_k^+(\cdot)$, respectively, for task $k = 1, \ldots, K$. The two networks have the same given basis functions $\phi(\cdot) = [1, \phi^1(\cdot), \cdots, \phi^N(\cdot)]^T$, but different hidden-to-output weights. The weights of $f_k(\cdot)$ are trained with $\mathcal{D}_k$, while the weights of $f_k^+(\cdot)$ are trained using $\mathcal{D}_k^+ = \mathcal{D}_k \cup \{(\mathbf{x}_{J_k+1,k}, y_{J_k+1,k})\}$. Then for $k = 1, \cdots, K$, the squared errors committed on $(\mathbf{x}_{J_k+1,k}, y_{J_k+1,k})$ by $f_k(\cdot)$ and $f_k^+(\cdot)$ are related by $\left[f_k^+(\mathbf{x}_{J_k+1,k}) - y_{J_k+1,k}\right]^2 = \left[\gamma(\mathbf{x}_{J_k+1,k})\right]^{-1}\left[f_k(\mathbf{x}_{J_k+1,k}) - y_{J_k+1,k}\right]^2$, where $\gamma(\mathbf{x}_{J_k+1,k}) = \left[1 + \phi^T(\mathbf{x}_{J_k+1,k})\mathbf{A}_k^{-1}\phi(\mathbf{x}_{J_k+1,k})\right]^2 \geq 1$ and $\mathbf{A}_k = \sum_{i=1}^{J_k}\left[\rho\mathbf{I} + \phi(\mathbf{x}_{ik})\phi^T(\mathbf{x}_{ik})\right]$ is the same as in (3).*

Two observations are made from Corollary 2. First, if $\gamma(\mathbf{x}_{J_k+1,k}) \approx 1$, seeing $y_{J_k+1,k}$ does not effect the error on $\mathbf{x}_{J_k+1,k}$, indicating $\mathcal{D}_k$ already contain sufficient information about $(\mathbf{x}_{J_k+1,k}, y_{J_k+1,k})$. Second, if $\gamma(\mathbf{x}_i) \gg 1$, seeing $y_{J_k+1,k}$ greatly decrease the error on $\mathbf{x}_{J_k+1,k}$, indicating $\mathbf{x}_{J_k+1,k}$ is significantly dissimilar (novel) to $\mathcal{D}_k$ and $\mathbf{x}_{J_k+1,k}$ must be supervised to reduce the error. Based on Corollary 2, the selection proceeds sequentially. Suppose we have selected data $\mathcal{D}_k = \{(\mathbf{x}_{ik}, y_{ik})\}_{i=1}^{J_k}$, from which we compute $\mathbf{A}_k$. We select the next data point as $\mathbf{x}_{J_k+1,k} = \arg\max_{i>J_k, k=1,\cdots,K} \gamma(\mathbf{x}_{ik}) = \arg\max_{i>J_k, k=1,\cdots,K} \left[1 + \phi^T(\mathbf{x}_{ik})\mathbf{A}_k^{-1}\phi(\mathbf{x}_{ik})\right]^2$. After $\mathbf{x}_{J_k+1,k}$ is selected, the $\mathbf{A}_k$ is updated and the next selection begins. As the iteration advances $\gamma$ will decrease until it reaches convergence. We use (3) to compute $\mathbf{w}$ from the selected $\mathbf{x}$ and their associated targets $y$, completing learning of the RBF network.

## 5 Experimental Results

In this section we compare the multi-task RBF network against single-task RBF networks via experimental studies. We consider three types of RBF networks to learn $K$ tasks, each

with its data set $\mathcal{D}_k$. In the first, which we call "one RBF network", we let the $K$ tasks share both basis functions $\phi$ (hidden nodes) and hidden-to output weights $\mathbf{w}$, thus we do not distinguish the $K$ tasks and design a single RBF network to learn a union of them. The second is the multi-task RBF network, where the $K$ tasks share the same $\phi$ but each has its own $\mathbf{w}$. In the third, we have $K$ independent networks, each designed for a single task.

We use a school data set from the Inner London Education Authority, consisting of examination records of 15362 students from 139 secondary schools. The data are available at http://multilevel.ioe.ac.uk/intro/datasets.html. This data set was originally used to study the effectiveness of schools and has recently been used to evaluate multi-task algorithms [2,3]. The goal is to predict the exam scores of the students based on 9 variables: year of exam (1985, 1986, or 1987), school code (1-139), FSM (percentage of students eligible for free school meals), VR1 band (percentage of students in school in VR band one), gender, VR band of student (3 categories), ethnic group of student (11 categories), school gender (male, female, or mixed), school denomination (3 categories). We consider each school a task, leading to 139 tasks in total. The remaining 8 variables are used as inputs to the RBF network. Following [2,3], we converted each categorical variable to a number of binary variables, resulting in a total number of 27 input variables, i.e., $\mathbf{x} \in \mathbb{R}^{27}$. The exam score is the target to be predicted.

The three types of RBF networks as defined above are designed as follows. The multi-task RBF network is implemented as the structure as shown in Figure 1 and trained with the learning algorithm in Table 1. The "one RBF network" is implemented as a special case of Figure 1, with a single output node and trained using the union of supervised data from all 139 schools. We design 139 independent RBF networks, each of which is implemented with a single output node and trained using the supervised data from a single school. We use the Gaussian RBF $\phi^n(\mathbf{x}) = \exp(-\frac{||\mathbf{x}-\mathbf{c}_n||^2}{2\sigma^2})$, where the $\mathbf{c}_n$'s are selected from training data points and $\sigma_n$'s are initialized as 20 and optimized as described in Table 1. The main role of the regularization parameter $\rho$ is to prevent the $\mathbf{A}$ matrices from being singular and it does not affect the results seriously. In the results reported here, $\rho$ is set to $10^{-6}$.

Following [2-3], we randomly take 75% of the 15362 data points as training (supervised) data and the remaining 25% as test data. The generalization performance is measured by the squared error $(f_k(\mathbf{x}_{ik}) - y_{ik})^2$ averaged over all test data $\mathbf{x}_{ik}$ of tasks $k = 1, \cdots, K$. We made 10 independent trials to randomly split the data into training and test sets and the squared error averaged over the test data of all the 139 schools and the trials are shown in Table 2, for the three types of RBF networks.

Table 2: Squared error averaged over the test data of all 139 schools and the 10 independent trials for randomly splitting the school data into training (75%) and testing (25%) sets.

| Multi-task RBF network | Independent RBF networks | One RBF network |
|---|---|---|
| $109.89 \pm 1.8167$ | $136.41 \pm 7.0081$ | $149.48 \pm 2.8093$ |

Table 2 clearly shows the multi-task RBF network outperforms the other two types of RBF networks by a considerable margin. The "one RBF network" ignores the difference between the tasks and the independent RBF networks ignore the tasks' correlations, therefore they both perform inferiorly. The multi-task RBF network uses the shared hidden nodes (basis functions) to capture the common internal representation of the tasks and meanwhile uses the independent hidden-to-output weights to learn the statistics specific to each task.

We now demonstrate the results of active learning. We use the method in Section 4 to actively split the data into training and test sets using a two-step procedure. First we learn the basis functions $\phi$ of multi-task RBF network using all 15362 data (unsupervised). Based on the $\phi$, we then select the data to be supervised and use them as training data to learn

the hidden-to-output weights $\mathbf{w}$. To make the results comparable, we use the same training data to learn the other two types of RBF networks (including learning their own $\phi$ and $\mathbf{w}$). The networks are then tested on the remaining data.

Figure 2 shows the results of active learning. Each curve is the squared error averaged over the test data of all 139 schools, as a function of number of training data. It is clear that the multi-task RBF network maintains its superior performance all the way down to 5000 training data points, whereas the independent RBF networks have their performances degraded seriously as the training data diminish. This demonstrates the increasing advantage of multi-task learning as the number of training data decreases. The "one RBF network" seems also insensitive to the number of training data, but it ignores the inherent dissimilarity between the tasks, which makes its performance inferior.

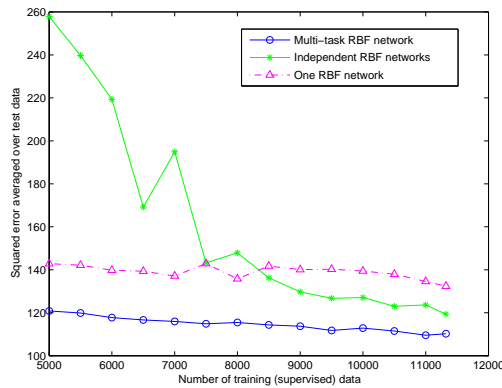

Figure 2: Squared error averaged over the test data of all 139 schools, as a function of the number of training (supervised) data. The data are split into training and test sets via active learning.

## 6   Conclusions

We have presented the structure and learning algorithms for multi-task learning with the radial basis function (RBF) network. By letting multiple tasks share the basis functions (hidden nodes) we impose a common internal representation for correlated tasks. Exploiting the inter-task correlation yields a more compact network structure that has enhanced generalization ability. Unsupervised learning of the network structure enables us to actively split the data into training and test sets. As the data novel to the previously selected ones are selected next, what finally remain unselected and to be tested are all similar to the selected data which constitutes the training set. This improves the generalization of the resulting network to the test data. These conclusions are substantiated via results on real multi-task data.

### References

[1] R. Caruana. (1997) Multitask learning. *Machine Learning*, 28, p. 41-75, 1997.

[2] B. Bakker and T. Heskes (2003). Task clustering and gating for Bayesian multitask learning. *Journal of Machine Learning Research*, 4: 83-99, 2003

[3] T. Evgeniou, C. A. Micchelli, and M. Pontil (2005). Learning Multiple Tasks with Kernel Methods. *Journal of Machine Learning Research*, 6: 615637, 2005

[4] Powell M. (1987), Radial basis functions for multivariable interpolation : A review, J.C. Mason and M.G. Cox, eds, *Algorithms for Approximation*, pp.143-167.

[5] Chen, F. Cowan, and P. Grant (1991), Orthogonal least squares learning algorithm for radial basis function networks, *IEEE Transactions on Neural Networks*, Vol. 2, No. 2, 302-309, 1991

[6] Cohn, D. A., Ghahramani, Z., and Jordan, M. I. (1995). Active learning with statistical models. *Advances in Neural Information Processing Systems*, 7, 705-712.

[7] V. Fedorov (1972), *Theory of Optimal Experiments*, Academic Press, 1972

[8] M. Stone (1974), Cross-validatory choice and assessment of statistical predictions, *Journal of the Royal Statistical Society, Series B*, 36, pp. 111-147, 1974.

## Appendix

**Proof of Theorem 1:**. Let $\phi^{new} = [\phi, \phi^{N+1}]^T$. By (3), the $\mathbf{A}$ matrices corresponding to $\phi^{new}$ are

$$\mathbf{A}_k^{new} = \sum_{i=1}^{J_k} \begin{bmatrix} \phi_{ik} \\ \phi_{ik}^{N+1} \end{bmatrix} \begin{bmatrix} \phi_{ik}^T & \phi_{ik}^{N+1} \end{bmatrix} + \rho\,\mathbf{I}_{(N+2)\times(N+2)} = \begin{bmatrix} \mathbf{A}_k & \mathbf{c}_k \\ \mathbf{c}_k^T & d_k \end{bmatrix} \qquad (A\text{-}1)$$

where $\mathbf{c}_k$ and $d_k$ are as in (6). By the conditions of the theorem, the matrices $\mathbf{A}_k$ and $\mathbf{A}_k^{new}$ are all non-degenerate. Using the block matrix inversion formula [7] we get

$$(\mathbf{A}_k^{new})^{-1} = \begin{bmatrix} \mathbf{A}_k^{-1} + \mathbf{A}_k^{-1}\mathbf{c}_k q_k^{-1}\mathbf{c}_k^T\mathbf{A}_k^{-1} & -\mathbf{A}_k^{-1}\mathbf{c}_k q_k^{-1} \\ -q_k^{-1}\mathbf{c}_k^T\mathbf{A}_k^{-1} & q_k^{-1} \end{bmatrix} \qquad (A\text{-}2)$$

where $q_k$ is as in (6). By (3), the weights $\mathbf{w}_k^{new}$ corresponding to $[\phi^T, \phi^{N+1}]^T$ are

$$\mathbf{w}_k^{new} = (\mathbf{A}_k^{new})^{-1} \begin{bmatrix} \sum_{i=1}^{J_k} y_{ik}\phi_{ik} \\ \sum_{i=1}^{J_k} y_{ik}\phi_{ik}^{N+1} \end{bmatrix} = \begin{bmatrix} \mathbf{w}_k + \mathbf{A}_k^{-1}\mathbf{c}_k q_k^{-1} g_k \\ -q_k^{-1} g_k \end{bmatrix} \qquad (A\text{-}3)$$

with $g_k = \mathbf{c}_k^T\mathbf{w}_k - \sum_{i=1}^{J_k} y_{ik}\phi_{ik}^{N+1}$. Hence, $(\phi_{ik}^{new})^T\mathbf{w}_k^{new} = \phi_{ik}^T\mathbf{w}_k + (\phi_{ik}^T\mathbf{A}_k^{-1}\mathbf{c}_k - \phi_{ik}^{N+1})g_k q_k^{-1}$, which is put into (4) to get $e(\phi^{new} = \sum_{k=1}^{K}\sum_{i=1}^{J_k}\left[y_{ik}^2 - y_{ik}(\phi_{ik}^{new})^T\mathbf{w}_k^{new}\right] = \sum_{k=1}^{K}\sum_{i=1}^{J_k}\left[y_{ik}^2 - y_{ik}\phi_{ik}^T\mathbf{w}_k - y_{ik}(\phi_{ik}^T\mathbf{A}_k^{-1}\mathbf{c}_k - \phi_{ik}^{N+1})g_k q_k^{-1}\right] = e(\phi) - \sum_{k=1}^{K}(\mathbf{c}_k^T\mathbf{w}_k - \sum_{i=1}^{J_k} y_{ik}\phi_{ik}^{N+1})^2 q_k^{-1}$, where in arriving the last equality we have used (3) and (4) and $g_k = \mathbf{c}_k^T\mathbf{w}_k - \sum_{i=1}^{J_k} y_{ik}\phi_{ik}^{N+1}$. The theorem is proved. $\qquad\square$

**Proof of Theorem 2:** The proof applies to $k = 1, \cdots, K$. For any given $k$, define $\mathbf{\Phi} = \left[\phi(\mathbf{x}_{1k}), \ldots, \phi(\mathbf{x}_{J_kk})\right]$, $\widetilde{\mathbf{\Phi}} = \left[\phi(\mathbf{x}_{J_k+1,k}), \ldots, \phi(\mathbf{x}_{J_k+\tilde{J}_k,k})\right]$, $\mathbf{y}_k = [y_{1k}, \ldots, y_{J_kk}]^T$, $\widetilde{\mathbf{y}}_k = [y_{J_k+1,k}, \ldots, y_{J_k+\tilde{J}_k,k}]^T$, $\mathbf{f}_k = [f(\mathbf{x}_{1k}), \ldots, f(\mathbf{x}_{J_kk})]^T$, $\mathbf{f}_k^{\sim} = [f_k^{\sim}(\mathbf{x}_{1k}), \ldots, f_k^{\sim}(\mathbf{x}_{J_kk})]^T$, and $\widetilde{\mathbf{A}}_k = \rho\mathbf{I} + \widetilde{\mathbf{\Phi}}_k\widetilde{\mathbf{\Phi}}_k^T$. By (1), (3), and the conditions of the theorem, $\mathbf{f}_k = \mathbf{\Phi}_k^T(\widetilde{\mathbf{A}}_k + \mathbf{\Phi}_k\mathbf{\Phi}_k^T)^{-1}(\mathbf{\Phi}_k\mathbf{y}_k + \widetilde{\mathbf{\Phi}}\widetilde{\mathbf{y}}_k) \overset{(a)}{=} \left[\mathbf{\Phi}_k^T\widetilde{\mathbf{A}}_k^{-1} - (\mathbf{\Phi}_k^T\widetilde{\mathbf{A}}_k^{-1}\mathbf{\Phi}_k + \mathbf{I} - \mathbf{I})(\mathbf{I} + \mathbf{\Phi}_k^T\widetilde{\mathbf{A}}_k^{-1}\mathbf{\Phi}_k)^{-1}\mathbf{\Phi}_k^T\widetilde{\mathbf{A}}_k^{-1}\right]\left[\mathbf{\Phi}_k\mathbf{y}_k + \widetilde{\mathbf{\Phi}}_k\widetilde{\mathbf{y}}_k\right] = \left[(\mathbf{I} + \mathbf{\Phi}_k^T\widetilde{\mathbf{A}}_k^{-1}\mathbf{\Phi}_k)^{-1}\mathbf{\Phi}_k^T\widetilde{\mathbf{A}}_k^{-1}\right]\left[\widetilde{\mathbf{\Phi}}_k\widetilde{\mathbf{y}}_k + \mathbf{\Phi}_k\mathbf{y}_k\right] \overset{(b)}{=} (\mathbf{I} + \mathbf{\Phi}_k^T\widetilde{\mathbf{A}}_k^{-1}\mathbf{\Phi}_k)^{-1}\mathbf{f}_k^{\sim} + (\mathbf{I} + \mathbf{\Phi}_k^T\widetilde{\mathbf{A}}_k^{-1}\mathbf{\Phi}_k)^{-1}(\mathbf{\Phi}_k^T\widetilde{\mathbf{A}}_k^{-1}\mathbf{\Phi}_k + \mathbf{I} - \mathbf{I})\mathbf{y}_k = \mathbf{y}_k + (\mathbf{I} + \mathbf{\Phi}_k^T\widetilde{\mathbf{A}}_k^{-1}\mathbf{\Phi}_k)^{-1}(\mathbf{f}_k^{\sim} - \mathbf{y}_k)$, where equation $(a)$ is due to the Sherman-Morrison-Woodbury formula and equation $(b)$ results because $\mathbf{f}_k^{\sim} = \mathbf{\Phi}_k^T\widetilde{\mathbf{A}}_k^{-1}\widetilde{\mathbf{\Phi}}_k\widetilde{\mathbf{y}}_k$. Hence, $\mathbf{f}_k - \mathbf{y}_k = (\mathbf{I} + \mathbf{\Phi}_k^T\widetilde{\mathbf{A}}_k^{-1}\mathbf{\Phi}_k)^{-1}(\mathbf{f}_k^{\sim} - \mathbf{y}_k)$, which gives

$$\sum_{i=1}^{J_k}(y_{ik} - f_k(\mathbf{x}_{ik}))^2 = (\mathbf{f}_k - \mathbf{y}_k)^T(\mathbf{f}_k - \mathbf{y}_k) = (\mathbf{f}_k^{\sim} - \mathbf{y}_k)^T\mathbf{\Gamma}_k^{-1}(\mathbf{f}_k^{\sim} - \mathbf{y}_k) \qquad (A\text{-}4)$$

where $\mathbf{\Gamma}_k = \left[\mathbf{I} + \mathbf{\Phi}_k^T\widetilde{\mathbf{A}}_k^{-1}\mathbf{\Phi}_k\right]^2 = \left[\mathbf{I} + \mathbf{\Phi}_k^T(\rho\mathbf{I} + \widetilde{\mathbf{\Phi}}_k\widetilde{\mathbf{\Phi}}_k^T)^{-1}\mathbf{\Phi}_k\right]^2$.

By construction, $\mathbf{\Gamma}_k$ has all its eigenvalues no less than 1, i.e., $\mathbf{\Gamma}_k = \mathbf{E}_k^T\text{diag}[\lambda_{1k}, \cdots, \lambda_{J_kk}]\mathbf{E}_k$ with $\mathbf{E}_k^T\mathbf{E}_k = \mathbf{I}$ and $\lambda_{1k}, \cdots, \lambda_{J_kk} \geq 1$, which makes the first, second, and last inequality in (7) hold. Using this expansion of $\mathbf{\Gamma}_k$ in (A-4) we get

$$\sum_{i=1}^{J_k}(f_k(\mathbf{x}_{ik}) - y_{ik})^2 = (\mathbf{f}_k^{\sim} - \mathbf{y}_k)^T\mathbf{E}_k^T\text{diag}[\sigma_{1k}^{-1}, \ldots, \sigma_{J_kk}^{-1}](\mathbf{f}_k^{\sim} - \mathbf{y}_k)$$

$$\leq (\mathbf{f}_k^{\sim} - \mathbf{y}_k)^T\mathbf{E}_k^T[\lambda_{min,k}^{-1}\mathbf{I}]\mathbf{E}_k(\mathbf{f}_k^{\sim} - \mathbf{y}_k) = \lambda_{min,k}^{-1}\sum_{i=1}^{J_k}(f_k^{\sim}(\mathbf{x}_{ik}) - y_{ik})^2 \quad (A\text{-}5)$$

where the inequality results because $\lambda_{min,k} = \min(\lambda_{1,k}, \cdots, \lambda_{J_k,k})$. From (A-5) follows the fourth inequality in (7). The third inequality in (7) can be proven in in a similar way. $\qquad\square$